# Convex Methods for Transduction

**Tijl De Bie**
ESAT-SCD/SISTA, K.U.Leuven
Kasteelpark Arenberg 10
3001 Leuven, Belgium
tijl.debie@esat.kuleuven.ac.be

**Nello Cristianini**
Department of Statistics, U.C.Davis
360 Kerr Hall One Shields Ave.
Davis, CA-95616
nello@support-vector.net

## Abstract

The 2-class transduction problem, as formulated by Vapnik [1], involves finding a separating hyperplane for a labelled data set that is also maximally distant from a given set of unlabelled test points. In this form, the problem has exponential computational complexity in the size of the working set. So far it has been attacked by means of integer programming techniques [2] that do not scale to reasonable problem sizes, or by local search procedures [3].

In this paper we present a relaxation of this task based on semi-definite programming (SDP), resulting in a convex optimization problem that has polynomial complexity in the size of the data set.

The results are very encouraging for mid sized data sets, however the cost is still too high for large scale problems, due to the high dimensional search space. To this end, we restrict the feasible region by introducing an approximation based on solving an eigenproblem. With this approximation, the computational cost of the algorithm is such that problems with more than 1000 points can be treated.

## 1 Introduction

The general transduction task is the following: given a *training set* of labelled data, and a *working set* of unlabelled data (also called transduction samples), estimate the value of a classification function at the given points in the working set. Statistical learning results [1] suggest that this setting should deliver better results than the traditional 'inductive' setting, where a function needs to be learned first and only later tested on a test set of points chosen after the learning has been completed. Different algorithms have been proposed so far to take advantage of this advance knowledge of the test points (such as in [1], [2], [3], [4], [5], [6] and others).

Given this general task, much research has been concentrated on a specific approach to transduction (first proposed by Vapnik [1]), based on the use of Support Vector Machines (SVM's). In this case, the algorithm is aimed at finding a separating hyperplane for the training set that is also maximally distant from the (unlabelled) working set. This hyperplane is used to predict the labels for the working set points.

In this form, the problem has exponential computational complexity, and several approaches have been attempted to solve it. Generally they involve some form of

local search [3], or integer programming methods [2].

A recent development of convex optimization theory is Semi Definite Programming (SDP), a branch of that field aimed at optimizing over the cone of semi positive definite (SPD) matrices. One of its main attractions is that it has proven successful in constructing tight convex relaxations of hard combinatorial optimization problems [7]. SDP has recently been applied successfully to machine learning problems [8].

In this paper we show how to relax the problem of transduction into an SDP problem, that can then be solved by (polynomial time) convex optimization methods. Empirical results on mid-sized data sets are very promising, however, due to the dimensionality of the feasible region of the relaxed parameters, still the algorithm complexity appears too large to tackle large scale problems. Therefore, we subsequently shrink the feasible region by making an approximation that is based on a spectral clustering method. Positive empirical results will be given.

**Formal definition: transductive SVM.** Based on the dual of the 1-norm soft margin SVM with zero bias[1], the dual formulation of the transductive SVM optimization problem can be written as a minimization of the dual SVM cost function (which is the inverse margin plus training errors) over label matrix $\mathbf{\Gamma}$ ([1], p. 437):

$$\min_{\mathbf{\Gamma}} \max_{\boldsymbol{\alpha}} \quad 2\boldsymbol{\alpha}'\mathbf{e} - \boldsymbol{\alpha}'(\mathbf{K} \odot \mathbf{\Gamma})\boldsymbol{\alpha} \tag{1}$$

$$\text{s.t.} \quad C \geq \alpha_i \geq 0 \tag{2}$$

$$\mathbf{\Gamma} = \left( \begin{array}{c} \mathbf{y}^t \\ \mathbf{y}^w \end{array} \right) \cdot \left( \begin{array}{c} \mathbf{y}^t \\ \mathbf{y}^w \end{array} \right)' \tag{3}$$

$$y_i^w \in \{1, -1\} \tag{4}$$

The (symmetric) matrix $\mathbf{\Gamma}$ is thus parameterized by the unknown working set label vector $\mathbf{y}^w \in \{-1, 1\}^{n_w}$ (with $n_w$ the size of the working set). The vector $\mathbf{y}^t \in \{-1, 1\}^{n_t}$ (with $n_t$ the number of training points) is the given fixed vector containing the known labels for the training points. The (symmetric) matrix $\mathbf{K} \in \Re^{(n_w+n_t) \times (n_w+n_t)}$ is the entire kernel matrix on the training set together with the working set. The dual vector is denoted by $\boldsymbol{\alpha} \in \Re^{n_w+n_t}$, and $\mathbf{e}$ is a vector of appropriate size containing all ones. The symbol $\odot$ represents the elementwise matrix product. It is clear indeed this is a combinatorial problem. The computational complexity scales exponentially in the size of the working set.

**Further notation.** Scalars are lower case; vectors boldface lower case; matrices boldface upper case. The unit matrix is denoted by $\mathbf{I}$. A pseudo-inverse is denoted with a superscript $^\dagger$, a transpose with a $'$. For ease of notation, the training part of the label matrix (and thus also of the kernel matrix) is always assumed to be its upper $n_t \times n_t$ block (as is assumed already in (3)). Furthermore, the $n_{t+}$ positive training samples are assumed to correspond to the first entries in $\mathbf{y}^t$, the $n_{t-}$ negative samples being at the end of this vector.

## 2 Relaxation to an SDP problem

In this section, we will gradually derive a relaxed version of the transductive SVM formulation. To start with, we replace some of the constraints by an equivalent set:

**Proposition 2.1** *(3) and (4) are equivalent with the following set of constraints:*

$$[\mathbf{\Gamma}]_{i,j \in \{1:n_t, 1:n_t\}} = y_i^t y_j^t \tag{5}$$

$$
\begin{aligned}
diag(\mathbf{\Gamma}) &= \mathbf{e} && (6)\\
rank(\mathbf{\Gamma}) &= 1 && (7)
\end{aligned}
$$

The values of $\mathbf{\Gamma}$ will then indeed be equal to 1 or $-1$. It is basically the rank constraint that makes the resulting constrained optimization problem combinatorial.

Note that these constraints imply that $\mathbf{\Gamma}$ is semi positive definite (SPD): $\mathbf{\Gamma} \succeq \mathbf{0}$ (this follows trivially from (3), or from (6) together with (7)). Now, in literature (see eg [7]) it is observed that such an SPD rank one constraint can often be relaxed to only the SPD constraint without sacrificing too much of the performance. Furthermore:

**Proposition 2.2** *If we relax the constraints by replacing (7) with*

$$
\mathbf{\Gamma} \succeq \mathbf{0}, \tag{8}
$$

*the optimization problem becomes convex.*

This follows from the fact that $\mathbf{\Gamma}$ appears linearly in the cost function, and that the constraints (2), (5), (6) and (8) consist of only linear equalities and linear (matrix) inequalities in the variables. Further on it will be shown to be an SDP problem.

While this relaxation of the rank constraint makes the optimization problem convex, the result will not be a rank one matrix anymore; it will only provide an approximation for the optimal rank one matrix. Thus the values of $\mathbf{\Gamma}$ will not be equal to 1 or $-1$ anymore. However, it is well known that:

**Lemma 2.1** *A principal submatrix of an SPD matrix is also SPD [10].*

By applying this lemma on all $2 \times 2$ principal submatrices of $\mathbf{\Gamma}$, it is shown that

**Corollary 2.1** *From constraints (6) and (8) follows: $-1 \leq [\mathbf{\Gamma}]_{i,j} \leq 1$.*

This is the problem will solve here: optimize (1) subject to (2), (5), (6) and (8). In the remainder of this section we will reformulate the optimization problem into a standard form of SDP, make further simplifications based on the problem structure, and show how to extract an approximation for the labels from the result.

## 2.1 Formulation as a standard SDP problem

In the derivations in this subsection the equality constraints (5) and (6) will not be stated for brevity. Their consequences will be treated further in the paper. Furthermore, in the implementation, they will be enforced explicitly by the parameterization, thus they will not appear as constraints in the optimization problem. Also the SPD constraint (8) is not written every time, it should be understood.

Let $2\boldsymbol{\nu} \geq \mathbf{0}$ be the Lagrange dual variables corresponding to constraint $\alpha_i \geq 0$ and $2\boldsymbol{\mu} \geq \mathbf{0}$ corresponding to constraint $\alpha_i \leq C$. Then, since the problem is convex and thus the minimization and maximization are exchangeable (strong duality, see [8] for a brief introduction to duality), the optimization problem is equivalent with:

$$
\min_{\mathbf{\Gamma}, \boldsymbol{\nu} \geq \mathbf{0}, \boldsymbol{\mu} \geq \mathbf{0}} \max_{\boldsymbol{\alpha}} \quad 2\boldsymbol{\alpha}'(\mathbf{e} + \boldsymbol{\nu} - \boldsymbol{\mu}) - \boldsymbol{\alpha}'(\mathbf{K} \odot \mathbf{\Gamma})\boldsymbol{\alpha} + 2C\boldsymbol{\mu}'\mathbf{e}
$$

In case $\mathbf{K} \odot \mathbf{\Gamma}$ is rank deficient, $(\mathbf{e} + \boldsymbol{\nu} - \boldsymbol{\mu})$ will be orthogonal to the null space of $\mathbf{K} \odot \mathbf{\Gamma}$ (otherwise, the object function could grow to infinity, and this while $\boldsymbol{\nu}$ and $\boldsymbol{\mu}$ on the contrary are minimizing the objective). The maximum over $\boldsymbol{\alpha}$ is then reached for $\boldsymbol{\alpha} = (\mathbf{K} \odot \mathbf{\Gamma})^{\dagger}(\mathbf{e} + \boldsymbol{\nu} - \boldsymbol{\mu})$. Substituting this in the object function gives:

$$
\min_{\mathbf{\Gamma}, \boldsymbol{\nu} \geq \mathbf{0}, \boldsymbol{\mu} \geq \mathbf{0}} (\mathbf{e} + \boldsymbol{\nu} - \boldsymbol{\mu})'(\mathbf{K} \odot \mathbf{\Gamma})^{\dagger}(\mathbf{e} + \boldsymbol{\nu} - \boldsymbol{\mu}) + 2C\boldsymbol{\mu}'\mathbf{e}
$$

or equivalently:

$$\min_{\boldsymbol{\Gamma},\boldsymbol{\nu}\geq\mathbf{0},\boldsymbol{\mu}\geq\mathbf{0},t} t \quad \text{s.t.} \quad t \geq (\mathbf{e}+\boldsymbol{\nu}-\boldsymbol{\mu})'(\mathbf{K}\odot\boldsymbol{\Gamma})^{\dagger}(\mathbf{e}+\boldsymbol{\nu}-\boldsymbol{\mu})+2C\boldsymbol{\mu}'\mathbf{e}.$$

with as additional constraint that $(\mathbf{e}+\boldsymbol{\nu}-\boldsymbol{\mu})$ is orthogonal to the null space of $\mathbf{K}\odot\boldsymbol{\Gamma}$. This latter constraint and the quadratic constraint can be reformulated as one SPD constraint thanks to the following extension of the Schur complement lemma [10] (the proof is omitted due to space restrictions):

**Lemma 2.2 (Extended Schur complement lemma)** *For symmetric* $\mathbf{A} \succeq \mathbf{0}$ *and* $\mathbf{C} \succ \mathbf{0}$:

$$\left.\begin{array}{rcl} \textit{The column space of } \mathbf{B} & \perp & \textit{the null space of } \mathbf{A} \\ \mathbf{C} & \succeq & \mathbf{B}'\mathbf{A}^{\dagger}\mathbf{B} \end{array}\right\} \Leftrightarrow \left(\begin{array}{cc} \mathbf{A} & \mathbf{B} \\ \mathbf{B}' & \mathbf{C} \end{array}\right) \succeq \mathbf{0}.$$

Indeed, applying this lemma to our problem with $\mathbf{A} = \mathbf{K}\odot\boldsymbol{\Gamma}$, $\mathbf{B} = \mathbf{e}+\boldsymbol{\nu}-\boldsymbol{\mu}$ and $\mathbf{C} = t - 2C\boldsymbol{\mu}'\mathbf{e}$, leads to the problem formulation in the standard SDP form:

$$\min_{\boldsymbol{\Gamma},\boldsymbol{\nu}\geq\mathbf{0},\boldsymbol{\mu}\geq\mathbf{0},t} t \tag{9}$$

$$\text{s.t.} \quad \left(\begin{array}{cc} \mathbf{K}\odot\boldsymbol{\Gamma} & (\mathbf{e}+\boldsymbol{\nu}-\boldsymbol{\mu}) \\ (\mathbf{e}+\boldsymbol{\nu}-\boldsymbol{\mu})' & t - 2C\boldsymbol{\mu}'\mathbf{e} \end{array}\right) \succeq \mathbf{0} \tag{10}$$

together with the constraints (5), (6) and (8). The relaxation for the hard margin SVM is found by following a very similar derivation, or by just equating $\boldsymbol{\mu}$ to $\mathbf{0}$.

The number of variables specifying $\boldsymbol{\Gamma}$, and the size of constraint (8) can be greatly reduced due to structure in the problem. This is subject of what follows now.

## 2.2 Simplifications due to the problem structure

The matrix $\boldsymbol{\Gamma}$ can be parameterized as $\boldsymbol{\Gamma} = \left(\begin{array}{cc} \mathbf{y}^t\mathbf{y}^{t'} & \boldsymbol{\Gamma}^c \\ \boldsymbol{\Gamma}^{c'} & \boldsymbol{\Gamma}^w \end{array}\right)$ where we have a training block $\mathbf{y}^t\mathbf{y}^{t'} \in \Re^{n_t \times n_t}$, cross blocks $\boldsymbol{\Gamma}^c \in \Re^{n_t \times n_w}$ and $\boldsymbol{\Gamma}^{c'}$, and a transduction block $\boldsymbol{\Gamma}^w \in \Re^{n_w \times n_w}$, which is a symmetric matrix with diagonal entries equal to 1. We now use Lemma 2.1: by choosing a submatrix that contains all rows and columns corresponding to the training block, and just one row and column corresponding to the transduction part, the SPD constraint of $\boldsymbol{\Gamma}$ is seen to imply that

$$\boldsymbol{\Gamma} = \left(\begin{array}{cc} \mathbf{y}^t\mathbf{y}^{t'} & \boldsymbol{\gamma}_i^c \\ \boldsymbol{\gamma}_i^{c'} & 1 \end{array}\right) \succeq \mathbf{0}$$

where $\boldsymbol{\gamma}_i^c$ represents the $i$th column of $\boldsymbol{\Gamma}^c$. Using the extended Schur complement lemma 2.2, it follows that $\boldsymbol{\gamma}_i^c$ is proportional to $\mathbf{y}^t$ (denoted by $\boldsymbol{\gamma}_i^c = g_i\mathbf{y}^t$), and $1 \succeq \boldsymbol{\gamma}_i^{c'}\left(\mathbf{y}^t\mathbf{y}^{t'}\right)^{\dagger}\boldsymbol{\gamma}_i^c = \boldsymbol{\gamma}_i^{c'}\frac{\mathbf{y}^t\mathbf{y}^{t'}}{\|\mathbf{y}^t\|^4}\boldsymbol{\gamma}_i^c$. This implies that $1 \geq g_i\mathbf{y}^{t'}\frac{\mathbf{y}^t\mathbf{y}^{t'}}{\|\mathbf{y}^t\|^4}\mathbf{y}^t g_i = g_i^2$ such that $-1 \leq g_i \leq 1$. (Note that this is a corollary of the SPD constraint and does not need to be imposed explicitly.) Thus, the parameterization of $\boldsymbol{\Gamma}$ can be reduced to:

$$\boldsymbol{\Gamma} = \left(\begin{array}{cc} \mathbf{y}^t\mathbf{y}^{t'} & \mathbf{y}^t\mathbf{g}' \\ \mathbf{g}\mathbf{y}^{t'} & \boldsymbol{\Gamma}^w \end{array}\right) \quad \text{with} \quad \boldsymbol{\Gamma}_{ii}^w = 1$$

where $\mathbf{g}$ is the vector with $g_i$ as $i$th entry. We can now show that:

**Proposition 2.3** *The constraint* $\boldsymbol{\Gamma} \succeq \mathbf{0}$ *is equivalent to (and can thus be replaced by) the following SPD constraint on a smaller matrix* $\widetilde{\boldsymbol{\Gamma}}$:

$$\widetilde{\boldsymbol{\Gamma}} = \left(\begin{array}{cc} 1 & \mathbf{g}' \\ \mathbf{g} & \boldsymbol{\Gamma}^w \end{array}\right) \succeq \mathbf{0}.$$

Since $\widetilde{\boldsymbol{\Gamma}}$ is a principal submatrix of $\boldsymbol{\Gamma}$ (assuming at least one training label is equal to 1), lemma 2.1 indeed shows that $\boldsymbol{\Gamma} \succeq \boldsymbol{0}$ implies $\widetilde{\boldsymbol{\Gamma}} \succeq \boldsymbol{0}$. On the other hand, note that by adding a column and corresponding row to $\widetilde{\boldsymbol{\Gamma}}$, the rank is not increased. Thus, an eigenvalue equal to 0 is added. Due to the interlacing property for bordered matrices [10] and the fact that $\widetilde{\boldsymbol{\Gamma}} \succeq \boldsymbol{0}$, we know this can only be the smallest eigenvalue of the resulting matrix. By induction this shows that also $\widetilde{\boldsymbol{\Gamma}} \succeq \boldsymbol{0}$ implies $\boldsymbol{\Gamma} \succeq \boldsymbol{0}$.

This is the final formulation of the problem. For the soft margin case, the number of parameters is now $1+2n_t+\frac{n_w^2+5n_w}{2}$. For the hard margin case, this is $1+n_t+\frac{n_w^2+3n_w}{2}$.

### 2.3  Extraction of an estimate for the labels from $\boldsymbol{\Gamma}$

In general, the optimal $\boldsymbol{\Gamma}$ will of course not be rank one. We can approximate it by a rank one matrix however, by taking $\mathbf{g}$ as an approximation for the labels optimizing the unrelaxed problem. This is the approach we adopt: a thresholded value of the entries of $\mathbf{g}$ will be taken as a guess for the labels of the working set.

Note that the minimum of the relaxed problem is always smaller than or equal to the minimum of the unrelaxed problem. Furthermore, the minimum of the unrelaxed problem is smaller than or equal to the value achieved by the thresholded relaxed labels. Thus, we obtain a *lower* and an *upper bound* for the true optimal cost.

### 2.4  Remarks

The performance of this method is very good, as is seen on a toy problem (figure 1 shows an illustrative example). However, due to the (even though polynomial) complexity of SDP in combination with the quadratic dependence of the number of variables on the number of transduction points[2], problems with more than about 1000 training samples and 100 transduction samples can not practically be solved with general purpose SDP algorithms. Especially the limitation on the working set is a drawback, since the advantage of transduction becomes apparent especially for a large working set as compared to the number of training samples. This makes the applicability of this approach for large real life problems rather limited.

## 3  Subspace SDP formulation

However, if we would know a subspace (spanned by the $d$ columns of a matrix $\mathbf{V} \in \Re^{(n_t+n_w)\times d}$) in which (or close to which) the label vector lies, we can restrict the feasible region for $\boldsymbol{\Gamma}$, leading to a much more efficient algorithm. In the next section a fast method to estimate such a space $\mathbf{V}$ will be provided. In this section we assume $\mathbf{V}$ is known, and explain how to do the reduction of the feasible region.

If we know that the true label vector $\mathbf{y}$ lies in the column space of a matrix $\mathbf{V}$, we know the true label matrix can be written in the form $\boldsymbol{\Gamma} = \mathbf{V}\mathbf{M}\mathbf{V}'$, with $\mathbf{M}$ a symmetric matrix. The number of parameters is now only $d(d+1)/2$. Furthermore, constraint (8) that $\boldsymbol{\Gamma} \succeq 0$ is then equivalent to $\mathbf{M} \succeq 0$, which is a cheaper constraint.

Note however that in practical cases, the true label vector will not lie within but only close to the subspace spanned by the columns of $\mathbf{V}$. Then the diagonal of the label matrix $\boldsymbol{\Gamma}$ can not always be made exactly equal to $\mathbf{e}$ as required by (6). We thus relax this constraint to the requirement that the diagonal is not larger than

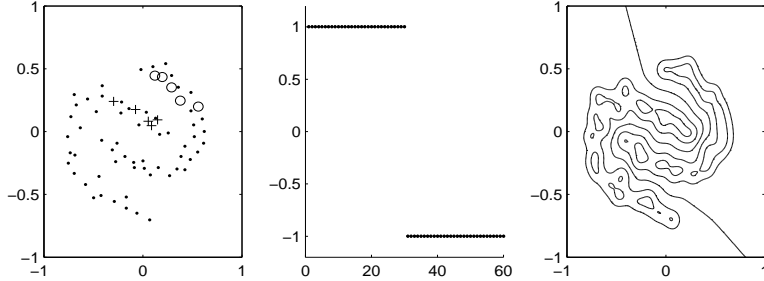

Figure 1: The left picture shows 10 labelled samples represented by a 'o' or a '+', depending on their class, together with 60 unlabelled samples represented by a '·'. The middle picture shows the labels for the working set as estimated using the SDP method before thresholding: all are already invisibly close to 1 or $-1$. The right picture shows contour lines of the classification surface obtained by training an SVM using all labels as found by the SDP method. The method clearly finds a visually good label assignment that takes cluster structure in the data into account.

**e**. Similarly, the block in the label matrix corresponding to the training samples may not contain 1's and $-1$'s exactly (constraint (5)). However, the better $\mathbf{V}$ is chosen, the better this constraint will be met. Thus we optimize (9) subject to (10) together with three constraints that replace the constraints (5), (6) and (8):

$$
\begin{aligned}
\mathbf{\Gamma} &= \mathbf{VMV}' \\
\operatorname{diag}(\mathbf{\Gamma}) &\leq \mathbf{e} \\
\mathbf{M} &\succeq \mathbf{0}
\end{aligned}
$$

Thus we can approximate the relaxed transductive SVM using this reduced parameterization for $\mathbf{\Gamma}$. The number of effective variables is now only a linear function of $n_w$: $1 + n_t + n_w + d(d+1)/2$ for a hard margin and $1 + 2(n_t + n_w) + d(d+1)/2$ for a soft margin SVM. Furthermore, one of the SPD constraints is now a constraint on a $d \times d$ matrix instead of a potentially large $(n_w + 1) \times (n_w + 1)$ matrix. For a constant $d$, the worst case complexity is thus reduced to $O((n_t + n_w)^{4.5})$.

The quality of the approximation can be determined by the user: the number of components $d$ can be chosen depending on the available computing resources, however empirical results show a good performance already for relatively small $d$.

## 4  Spectral transduction to find the subspace

In this section we will discuss how to find a subspace $\mathbf{V}$ close to which the label vector will lie. Our approach is based on the spectral clustering algorithm proposed in [11]. They start with computing the eigenvectors corresponding to the largest eigenvalues of $\mathbf{D}^{-1/2}\mathbf{K}\mathbf{D}^{-1/2}$ where $\mathbf{d} = \mathbf{Ke}$ contains all row sums of $\mathbf{K}$, and $\mathbf{D} = \operatorname{diag}(\mathbf{d})$. The dominant eigenvectors are shown to reflect the cluster structure of the data. The optimization problem corresponding to this eigenvalue problem is:

$$
\max_{\mathbf{v}} \quad \mathbf{v}'\mathbf{D}^{-1/2}\mathbf{K}\mathbf{D}^{-1/2}\mathbf{v} = \mathbf{v}'\widetilde{\mathbf{K}}\mathbf{v} \quad \text{s.t.} \quad \mathbf{v}'\mathbf{v} = \mathbf{1}. \tag{11}
$$

### 4.1  Constrained spectral clustering

We could apply this algorithm to the kernel matrix $\mathbf{K}$, but we can do more since we already know some of the labels: we will constrain the estimates of the labels

for the training samples that are known to be in the same class to be equal to each other. Then we optimize the same object function with respect to these additional constraints. This can be achieved by choosing the following parameterization for $\mathbf{v}$:

$$\mathbf{v} = \begin{pmatrix} \mathbf{e}_{n_{t+}}/\sqrt{n_{t+}} & \mathbf{0} & \mathbf{0} \\ \mathbf{0} & \mathbf{e}_{n_{t-}}/\sqrt{n_{t-}} & \mathbf{0} \\ \mathbf{0} & \mathbf{0} & \mathbf{I} \end{pmatrix} \cdot \begin{pmatrix} h^{t+} \\ h^{t-} \\ \mathbf{h}^w \end{pmatrix} = \mathbf{Lh}$$

where $\mathbf{e}_{n+}$ and $\mathbf{e}_{n-}$ denote the vectors containing $n_{t+}$ (the number of positive training samples) and $n_{t-}$ (the number of negative training samples) ones. Then:

**Proposition 4.1** *Optimization problem (11) is equivalent with:*

$$\max_{\mathbf{h}} \quad \mathbf{h}'\mathbf{L}'\mathbf{D}^{-1/2}\mathbf{K}\mathbf{D}^{-1/2}\mathbf{Lh} \quad s.t. \quad \mathbf{h}'\mathbf{h} = 1$$

*which corresponds to the eigenvalue problem $\mathbf{L}'\mathbf{D}^{-1/2}\mathbf{K}\mathbf{D}^{-1/2}\mathbf{Lh} = \lambda\mathbf{h}$. Then $\mathbf{v}$ is found as $\mathbf{v} = \mathbf{Lh}$.*

This is an extension of spectral clustering towards transduction[3]. We will use a subscript $i$ to denote the $i$th eigenvector and eigenvalue, where $\lambda_i \geq \lambda_j$ for $i > j$.

## 4.2 Spectral transduction provides a good V

By construction, all entries of $\mathbf{v}_i$ corresponding to positive training samples will be equal to $h_i^{t+}/\sqrt{n_{t+}}$; entries corresponding to the negative ones will all be equal to $h_i^{t-}/\sqrt{n_{t-}}$. Furthermore, as in spectral clustering, the other entries of vectors $\mathbf{v}_i$ with large eigenvalue $\lambda_i$ will reflect the cluster structure of the entire data set, while respecting the label assignment of the training points however[4]. This means that such a $\mathbf{v}_i$ will provide a good approximation for the labels. More specifically, the label vector will lie close to the column space of $\mathbf{V}$, having $d$ dominant 'centered' $\mathbf{v}_i$ as its columns; the larger $d$, the better the approximation. The way we 'center' $\mathbf{v}_i$ is by adding a constant so that entries for positive training samples become equal to minus those for the negative ones. Since then the first $n_t$ columns of the resulting $\mathbf{\Gamma} = \mathbf{VMV}'$ will be equal up to a sign, we can adopt basically the same approach as in section 2.3 to guess the labels: pick and threshold the first column of $\mathbf{\Gamma}$.

## 5 Empirical results

To show the potential of the method, we extracted data from the USPS data set to form two classes. The positive class is formed by 100 randomly chosen samples representing a number 0, and 100 representing a 1; the negative class by 100 samples representing a 2 and 100 representing a 3. Thus, we have a balanced classification problem with two classes of each 200 samples. The training set is chosen to contain only 10 samples from each of both classes, and is randomly drawn but evenly distributed over the 4 numbers. We used a hard margin SVM with an rbf kernel with $\sigma = 7$ (which is equal to the average distance of the samples to their nearest neighbors, verified to be a good value for the induction as well as for the

transduction case). The average ROC-score (area under the ROC-curve) over 10 randomizations is computed, giving $0.75 \pm 0.03$ as average for the inductive SVM, and $0.959 \pm 0.03$ for the method developed in this paper (we chose $d = 4$). To illustrate the scalability of the method, and to show that a larger working set is effectively exploited, we used a similar setting (same training set size) but with 1000 samples and $d = 3$, giving an average ROC-score of $0.993 \pm 0.004$.

## 6  Conclusions

We developed a relaxation for the transductive SVM as first proposed by Vapnik. It is shown how this combinatorial problem can be relaxed to an SDP problem.

Unfortunately, the number of variables in combination with the complexity of SDP is too high for it to scale to significant problem sizes. Therefore we show how, based on a new spectral method, the feasible region of the variables can be shrinked, leading to an approximation for the original SDP method. The complexity of the resulting algorithm is much more favorable. Positive empirical results are shown.

**Acknowledgement**

Tijl De Bie is a Research Assistant with the Fund for Scientific Research – Flanders (F.W.O.–Vlaanderen).

## Footnotes

[1]We do not include a bias term since this would make the problem too non-convex. However this does not impair the result as is explained in [9].

[2]The worst case complexity for the problem at hand is $O((n_t+n_w^2)^2(n_t+n_w)^{2.5})$, which is of order 6.5 in the number of transduction points $n_w$.

[3]We want to point out that the spectral transduction on its own is empirically observed to significantly improve over standard spectral clustering algorithms, and compares favorably with a recently proposed [5] extension of spectral clustering towards transduction. Furthermore, as also in [5] the method can be generalized towards a method for clustering with side-information (where side-information consists of sets of points that are known to be co-clustered). Space restrictions do not permit us to go into this in the current paper.

[4]Note: to reduce the influence from outliers, large entries of the $\mathbf{v}_i$ can be thresholded.

## References

[1] V. N. Vapnik. *Statistical Learning Theory*. Springer, 1998.

[2] K. Bennett and A. Demiriz. Semi-supervised support vector machines. In M. S. Kearns, S. A. Solla, and D. A. Cohn, editors, *Advances in Neural Information Processing Systems 11*, Cambridge, MA, 1999. MIT Press.

[3] T. Joachims. Transductive inference for text classification using support vector machines. In *Proc. of the International Conference on Machine Learning (ICML)*, 1999.

[4] N. Cristianini, J. Kandola, A. Elisseeff, and J. Shawe-Taylor. *On optimizing kernel alignment*. Submitted for publication, 2003.

[5] S. D. Kamvar, D. Klein, and C. D. Manning. Spectral learning. In *Proc. of the International Joint Conference on Artificial Intelligence (IJCAI)*, 2003.

[6] O. Chapelle, J. Weston, and B. Schölkopf. Cluster kernels for semi-supervised learning. In S. Becker, S. Thrun, and K. Obermayer, editors, *Advances in Neural Information Processing Systems 15*, Cambridge, MA, 2003. MIT Press.

[7] C. Helmberg. *Semidefinite Programming for Combinatorial Optimization*. Habilitationsschrift, TU Berlin, January 2000. ZIB-Report ZR-00-34, Konrad-Zuse-Zentrum Berlin, 2000.

[8] G. Lanckriet, N. Cristianini, P. Bartlett, L. El Ghaoui, and M. I. Jordan. Learning the kernel matrix with semidefinite programming. *Journal of Machine Learning Research (JMLR)*, 5:27–72, 2004.

[9] T. Poggio, S. Mukherjee, R. Rifkin, A. Rakhlin, and A. Verri. b. In *Proceedings of the Conference on Uncertainty in Geometric Computations*, 2001.

[10] R. A. Horn and C. R. Johnson. *Matrix Analysis*. Cambridge University Press, 1985.

[11] J. Shi and J. Malik. Normalized cuts and image segmentation. *IEEE Transactions on Pattern Analysis and Machine Intelligence*, 22(8):888–905, 2000.
